# Predicting Dynamic Difficulty

**Olana Missura** and **Thomas Gärtner**
University of Bonn and Fraunhofer IAIS
Schloß Birlinghoven
52757 Sankt Augustin, Germany
{olana.missura,thomas.gaertner}@uni-bonn.de

## Abstract

Motivated by applications in electronic games as well as teaching systems, we investigate the problem of dynamic difficulty adjustment. The task here is to repeatedly find a game difficulty setting that is neither 'too easy' and bores the player, nor 'too difficult' and overburdens the player. The contributions of this paper are $(i)$ the formulation of difficulty adjustment as an online learning problem on partially ordered sets, $(ii)$ an exponential update algorithm for dynamic difficulty adjustment, $(iii)$ a bound on the number of wrong difficulty settings relative to the best static setting chosen in hindsight, and $(iv)$ an empirical investigation of the algorithm when playing against adversaries.

## 1 Introduction

While difficulty adjustment is common practise in many traditional games (consider, for instance, the handicap in golf or the handicap stones in go), the case for dynamic difficulty adjustment in electronic games has been made only recently [7]. Still, there are already many different, more or less successful, heuristic approaches for implementing it. In this paper, we formalise dynamic difficulty adjustment as a game between a *master* and a *player* in which the *master* tries to predict the most appropriate difficulty setting. As the *player* is typically a human with changing performance depending on many hidden factors as well as luck, no assumptions about the *player* can be made. The difficulty adjustment game is played on a partially ordered set which reflects the 'more difficult than'-relation on the set of difficulty settings. To the best of our knowledge, in this paper, we provide the first thorough theoretical treatment of dynamic difficulty adjustment as a prediction problem.

The contributions of this paper are: We formalise the learning problem of dynamic difficulty adjustment (in Section 2), propose a novel learning algorithm for this problem (in Section 4), and give a bound on the number of proposed difficulty settings that were not just right (in Section 5). The bound limits the number of mistakes the algorithm can make relative to the best static difficulty setting chosen in hindsight. For the bound to hold, no assumptions whatsoever need to be made on the behaviour of the player. Last but not least we empirically study the behaviour of the algorithm under various circumstances (in Section 6). In particular, we investigate the performance of the algorithm 'against' statistically distributed players by simulating the players as well as 'against' adversaries by asking humans to try to trick the algorithm in a simplified setting. Implementing our algorithm into a real game and testing it on real human players is left to future work.

## 2 Formalisation

To be able to theoretically investigate dynamic difficulty adjustment, we view it as a game between a *master* and a *player*, played on a partially ordered set modelling the 'more difficult than'-relation. The game is played in turns where each turn has the following elements:

1. the game *master* chooses a difficulty setting,
2. the *player* plays one 'round' of the game in this setting, and
3. the game *master* experiences whether the setting was 'too difficult', 'just right', or 'too easy' for the player.

The *master* aims at making as few as possible mistakes, that is, at choosing a difficulty setting that is 'just right' as often as possible. In this paper, we aim at developing an algorithm for the *master* with theoretical guarantees on the number of mistakes in the worst case while not making any assumptions about the *player*.

To simplify our analysis, we make the following, rather natural assumptions:

- the set of difficulty settings is finite and
- in every round, the (hidden) difficulty settings respect the partial order, that is,
    - no state that 'is more difficult than' a state which is 'too difficult' can be 'just right' or 'too easy' and
    - no state that 'is more difficult than' a state which is 'just right' can be 'too easy'.

Even with these natural assumptions, in the worst case, no algorithm for the *master* will be able to make even a single correct prediction. As we can not make any assumptions about the *player*, we will be interested in comparing our algorithm theoretically and empirically with the best statically chosen difficulty setting, as is commonly the case in online learning [3].

## 3  Related Work

As of today there exist a few commercial games with a well designed dynamic difficulty adjustment mechanism, but all of them employ heuristics and as such suffer from the typical disadvantages (being not transferable easily to other games, requiring extensive testing, etc). What we would like to have instead of heuristics is a universal mechanism for dynamic difficulty adjustment: An online algorithm that takes as an input (game-specific) ways to modify difficulty and the current player's in-game history (actions, performance, reactions, ...) and produces as an output an appropriate difficulty modification.

Both artificial intelligence researchers and the game developers community display an interest in the problem of automatic difficulty scaling. Different approaches can be seen in the work of R. Hunicke and V. Chapman [10], R. Herbich and T. Graepel [9], Danzi et al [7], and others. Since the perceived difficulty and the preferred difficulty are subjective parameters, the dynamic difficulty adjustment algorithm should be able to choose the "right" difficulty level in a comparatively short time for any particular player. Existing work in player modeling in computer games [14, 13, 5, 12] demonstrates the power of utilising the player models to create the games or in-game situations of high interest and satisfaction for the players.

As can be seen from these examples the problem of dynamic difficulty adjustment in video games was attacked from different angles, but a unifying and theoretically sound approach is still missing. To the best of our knowledge this work contains the first theoretical formalization of dynamic difficulty adjustment as a learning problem.

Under the assumptions described in Section 2, we can view the partially ordered set as a directed acyclic graph, at each round labelled by three colours (say, red, for 'too difficult' green for 'just right', and blue for 'too easy') such that

- for every directed path in the graph between two equally labelled vertices, all vertices on that path have the same colour and
- there is no directed path from a green vertex to a red vertex and none from a blue vertex to either a red or a green vertex.

The colours are allowed to change in each round as long as they obey the above rules. The *master*, i.e., the learning algorithm, does not see the colours but must point at a green vertex as often as

possible. If he points at a red vertex, he receives the feedback $-1$; if he points at a blue vertex, he receives the feedback $+1$.

This setting is related to learning directed cuts with membership queries. For learning directed cuts, i.e., monotone subsets, Gärtner and Garriga [8] provided algorithms and bounds for the case in which the labelling does not change over time. They then showed that the intersection between a monotone and an antimonotone subset in not learnable. This negative result is not applicable in our case, as the feedback we receive is more powerful. They furthermore showed that directed cuts are not learnable with traditional membership queries if the labelling is allowed to change over time. This negative result also does not apply to our case as the aim of the *master* is "only" to point at a green vertex as often as possible and as we are interested in a comparison with the best static vertex chosen in hindsight.

If we ignore the structure inherent in the difficulty settings, we will be in a standard multi-armed bandit setting [2]: There are $K$ arms, to which an unknown adversary assigns loss values on each iteration (0 to the 'just right' arms, 1 to all the others). The goal of the algorithm is to choose an arm on each iteration to minimize its overall loss. The difficulty of the learning problem comes from the fact that only the loss of the chosen arm is revealed to the algorithm. This setting was studied extensively in the last years, see [11, 6, 4, 1] and others. The standard performance measure is the so-called 'regret': The difference of the loss acquired by the learning algorithm and by the best static arm chosen in hindsight. The best known to-date algorithm that does not use any additional information is the Improved Bandit Strategy (called IMPROVEDPI in the following) [3]. The upper bound on its regret is of the order $\sqrt{KT\ln(T)}$, where $T$ is the amount of iterations. IMPROVEDPI will be the second baseline after the best static in hindsight (BSIH) in our experiments.

## 4  Algorithm

In this section we give an exponential update algorithm for predicting a vertex that corresponds to a 'just right' difficulty setting in a finite partially ordered set $(\mathcal{K}, \succ)$ of difficulty settings. The partial order is such that for $i, j \in \mathcal{K}$ we write $i \succ j$ if difficulty setting $i$ is 'more difficult than' difficulty setting $j$. The learning rate of the algorithm is denoted by $\beta$. The response that the *master* algorithm can observe $o_t$ is $+1$ if the chosen difficulty setting was 'too easy', 0 if it was 'just right', and $-1$ if it was 'too difficult'. The algorithm maintains a belief $w$ of each vertex being 'just right' and updates this belief if the observed response implies that the setting was 'too easy' or 'too difficult'.

---

**Algorithm 1** PARTIALLY-ORDERED-SET MASTER (POSM) for Difficulty Adjustment

---

**Require:** parameter $\beta \in (0, 1)$, $K$ difficulty Settings $\mathcal{K}$, partial order $\succ$ on $\mathcal{K}$, and a sequence of observations $o_1, o_2, \ldots$

1: $\forall k \in \mathcal{K}$ : let $w_1(k) = 1$
2: **for** each turn $t = 1, 2, \ldots$ **do**
3: $\quad \forall k \in \mathcal{K}$ : let $A_t(k) = \sum_{x \in \mathcal{K}: x \succeq k} w_t(x)$
4: $\quad \forall k \in \mathcal{K}$ : let $B_t(k) = \sum_{x \in \mathcal{K}: x \preceq k} w_t(x)$
5: $\quad$ PREDICT $k_t = \mathrm{argmax}_{k \in \mathcal{K}} \min\{B_t(k), A_t(k)\}$
6: $\quad$ OBSERVE $o_t \in \{-1, 0, +1\}$
7: $\quad$ **if** $o_t = +1$ **then**
8: $\quad\quad \forall k \in \mathcal{K}$ : let $w_{t+1}(k) = \begin{cases} \beta w_t(k) & \text{if } k \preceq k_t \\ w_t(x) & \text{otherwise} \end{cases}$
9: $\quad$ **end if**
10: $\quad$ **if** $o_t = -1$ **then**
11: $\quad\quad \forall k \in \mathcal{K}$ : let $w_{t+1}(k) = \begin{cases} \beta w_t(k) & \text{if } k \succeq k_t \\ w_t(x) & \text{otherwise} \end{cases}$
12: $\quad$ **end if**
13: **end for**

---

The main idea of Algorithm 1 is that in each round we want to make sure we can update as much belief as possible. The significance of this will be clearer when looking at the theory in the next section. To ensure it, we compute for each setting $k$ the belief 'above' $k$ as well as 'below' $k$ .

That is, $A_t$ in line 3 of the algorithm collects the belief of all settings that are known to be 'more difficult' and $B_t$ in line 4 of the algorithm collects the belief of all settings that are known to be 'less difficult' than $k$. If we observe that the proposed setting was 'too easy', that is, we should 'increase the difficulty', in line 8 we update the belief of the proposed setting as well as all settings easier than the proposed. If we observe that the proposed setting was 'too difficult', that is, we should 'decrease the difficulty', in line 11 we update the belief of the proposed setting as well as all settings more difficult than the proposed. The amount of belief that is updated for each mistake is thus equal to $B_t(k_t)$ or $A_t(k_t)$. To gain the most information independent of the observation and thus to achieve the best performance, we choose the $k$ that gives us the best worst case update $\min\{B_t(k), A_t(k)\}$ in line 5 of the algorithm.

## 5 Theory

We will now show a bound on the number of inappropriate difficulty settings that are proposed, relative to the number of mistakes the best static difficulty setting makes. We denote the number of mistakes of POSM until time $T$ by $m$ and the minimum number of times a statically chosen difficulty setting would have made a mistake until time $T$ by $M$. We denote furthermore the total amount of belief on the partially ordered set by $W_t = \sum_{k \in \mathcal{K}} w_t(k)$.

The analysis of the algorithm relies on the notion of a path cover of $\mathcal{K}$, i.e., a set of paths covering $\mathcal{K}$. A path is a subset of $\mathcal{K}$ that is totally ordered. A set of paths is covering $\mathcal{K}$ if the union of the paths is equal to $\mathcal{K}$. Any path cover can be chosen but the minimum path cover of $\mathcal{K}$ achieves the tightest bound. It can be found in time polynomial in $|\mathcal{K}|$ and its size is equal to the size of the largest antichain in $(\mathcal{K}, \succ)$. We denote the chosen set of paths by $\mathcal{C}$.

With this terminology, we are now ready to state the main result of our paper:

**Theorem 1.** *For the number of mistakes of* POSM*, it holds that:*

$$
m \le \left\lfloor \frac{\ln|\mathcal{K}| + M \ln 1/\beta}{\ln \frac{2|\mathcal{C}|}{2|\mathcal{C}|-1+\beta}} \right\rfloor .
$$

For all $c \in \mathcal{C}$ we denote the amount of belief on every chain by $W_t^c = \sum_{x \in c} w_t(x)$, the belief 'above' $k$ on $c$ by $A_t^c(k) = \sum_{x \in c: x \succeq k} w_t(x)$, and the belief 'below' $k$ on $c$ by $B_t^c(k) = \sum_{x \in c: x \preceq k} w_t(x)$. Furthermore, we denote the 'heaviest' chain by $c_t = \mathrm{argmax}_{c \in \mathcal{C}} W_t^c$.

Unless stated otherwise, the following statements hold for all $t$.

**Observation 1.1.** *To relate the amount of belief updated by* POSM *to the amount of belief on each chain observe that*

$$
\begin{aligned}
\max_{k \in \mathcal{K}} \min\{A_t(k), B_t(k)\} &= \max_{c \in \mathcal{C}} \max_{k \in c} \min\{A_t(k), B_t(k)\} \\
&\ge \max_{c \in \mathcal{C}} \max_{k \in c} \min\{A_t^c(k), B_t^c(k)\} \\
&\ge \max_{k \in c_t} \min\{A_t^{c_t}(k), B_t^{c_t}(k)\} .
\end{aligned}
$$

**Observation 1.2.** *As $c_t$ is the 'heaviest' among all chains and $\sum_{c \in \mathcal{C}} W_t^c \ge W_T$, it holds that $W_t^{c_t} \ge W_t/|\mathcal{C}|$.*

We will next show that for every chain, there is a difficulty setting for which it holds that: If we proposed that setting and made a mistake, we would be able to update at least half of the total weight of that chain.

**Proposition 1.1.** *For all $c \in \mathcal{C}$ it holds that*

$$
\max_{k \in c} \min\{A_t^c(k), B_t^c(k)\} \ge W_t^c/2 .
$$

*Proof.* We choose

$$
i = \mathrm{argmax}_{k \in c}\{B_t^c(k) \mid B_t^c(k) < W_t^c/2\}
$$

and

$$j = \operatorname*{argmin}_{k \in c}\{B_t^c(k) \mid B_t^c(k) \geq W_t^c/2\} \, .$$

This way, we obtain $i, j \in c$ for which $B_t^c(i) < W_t^c/2 \leq B_t^c(j)$ and which are consecutive, that is, $\nexists k \in c : i \prec k \prec j$. Such $i, j$ exist and are unique as $\forall x \in \mathcal{K} : w_t(x) > 0$. We then have $B_t^c(i) + A_t^c(j) = W_t^c$ and thus also $A_t^c(j) > W_t^c/2$. This immediatelly implies

$$W_t^c/2 \leq \min\{A_t^c(j), B_t^c(j)\} \leq \max_{k \in c} \min\{A_t^c(k), B_t^c(k)\} \, .$$

$\square$

**Observation 1.3.** *We use the previous proposition to show that in each iteration in which* POSM *proposes an inappropriate difficulty setting, we update at least a constant fraction of the total weight of the partially ordered set:*

$$\max_{k \in \mathcal{K}} \min\{A_t(k), B_t(k)\} \geq \max_{k \in c_t} \min\{A_t^{c_t}(k), B_t^{c_t}(k)\} \geq \frac{W_t^{c_t}}{2} \geq \frac{W_t}{2|\mathcal{C}|}$$

*Proof (of Theorem 1).* From the previous observations it follows that at each mistake we update at least a fraction of $1/(2|\mathcal{C}|)$ of the total weight and have at most a fraction of $(2|\mathcal{C}| - 1)/(2|\mathcal{C}|)$ which is not updated. This implies

$$W_{t+1} \leq \beta \cdot \frac{1}{2|\mathcal{C}|} W_t + \frac{2|\mathcal{C}| - 1}{2|\mathcal{C}|} W_t \leq \left[\frac{\beta}{2|\mathcal{C}|} + \frac{2|\mathcal{C}| - 1}{2|\mathcal{C}|}\right] W_t \, .$$

Applying this bound recursively, we obtain for time $T$

$$W_T \leq W_0 \left[\frac{\beta}{2|\mathcal{C}|} + \frac{2|\mathcal{C}| - 1}{2|\mathcal{C}|}\right]^m \leq |\mathcal{K}| \left[\frac{\beta}{2|\mathcal{C}|} + \frac{2|\mathcal{C}| - 1}{2|\mathcal{C}|}\right]^m \, .$$

As we only update the weight of a difficulty setting if the response implied that the algorithm made a mistake, $\beta^M$ is a lower bound on the weight of one difficulty setting and hence also $W_T \geq \beta^M$. Solving

$$\beta^M \leq |\mathcal{K}| \left[\frac{|\mathcal{C}| - 1}{2|\mathcal{C}|} + \frac{\beta}{2|\mathcal{C}|}\right]^m$$

for $m$, proves the theorem.

$\square$

Note, that this bound is similar to the bound for the full information setting [3] despite much weaker information being available in our case. The influence of $|\mathcal{C}|$ is the new ingredient that changes the behaviour of this bound for different partially ordered sets.

## 6 Experiments

We performed two sets of experiments: simulating a game against a stochastic environment, as well as using human players to provide our algorithm with a non-oblivious adversary. To evaluate the performance of our algorithm we have chosen two baselines. The first one is the best static difficulty setting in hindsight: it is a difficulty that a player would pick if she knew her skill level in advance and had to choose the difficulty only once. The second one is the IMPROVEDPI algorithm [3].

In the following we denote the subset of poset's vertices with the 'just right' labels the zero-zone (because in the corresponding loss vector their components are equal to zero). In both stochastic and adversarial scenario we consider two different settings: so called 'smooth' and 'non-smooth' one. The settings' names describe the way the zero-zone changes with time. In the 'non-smooth' setting we don't place any restrictions on it apart from its size, while in the 'smooth' setting the border of the zero-zone is allowed to move only by one vertex at a time. These two settings represent two extreme situations: one player changing her skills gradually with time is changing the zero-zone 'smoothly'; different players with different skills for each new challenge the game presents will make the zero-zone 'jump'. In a more realistic scenario the zero-zone would change 'smoothly' most of the time, but sometimes it would perform jumps.

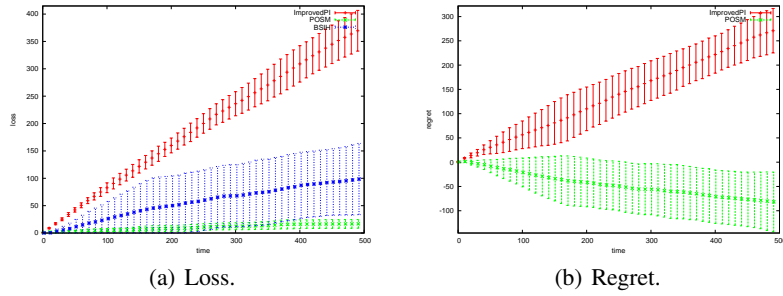

(a) Loss.　　　　　　　　　　　(b) Regret.

Figure 1: Stochastic adversary, 'smooth' setting, on a single chain of 50 vertices.

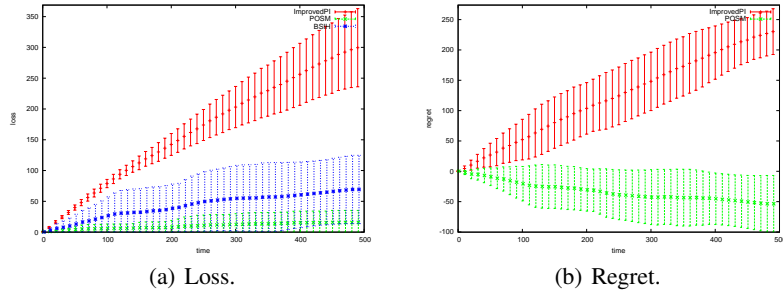

(a) Loss.　　　　　　　　　　　(b) Regret.

Figure 2: Stochastic adversary, 'smooth' setting, on a grid of 7x7 vertices.

## 6.1 Stochastic Adversary

In the first set of experiments we performed, the adversary is stochastic: On every iteration the zero-zone changes with a pre-defined probability. In the 'smooth' setting only one of the border vertices of the zero-zone at a time can change its label.For the 'non-smooth' setting we consider a truly evil case of limiting the zero-zone to always containing only one vertex and a case where the zero-zone may contain up to 20% of all the vertices in the graph. Note that even relabeling of a single vertex may break the consistency of the labeling with regard to the poset. The necessary repair procedure may result in more than one vertex being relabeled at a time.

We consider two graphs that represent two different but typical games structures with regard to the difficulty: a single chain and a 2-dimensional grid. A set of progressively more difficult challenges such that can be found in a puzzle or a time-management game can be directly mapped onto a chain of a length corresponding to the amount of challenges. A 2- (or more-) dimensional grid on the other hand is more like a skill-based game, where depending on the choices players make different game states become available to them. In our experiments the chain contains 50 vertices, while the grid is built on $7 \times 7$ vertices.

In all considered variations of the setting the game lasts for 500 iterations and is repeated 10 times. The resulting mean and standard deviation values of loss and regret, respectively, are shown in the following figures: The 'smooth' setting in Figures 1(a), 1(b) and 2(a), 2(b); The 'non-smooth' setting in Figures 3(a), 3(b) and 4(a), 4(b). (For brevity we omit the plots with the results of other 'non-smooth' variations. They all show very similar behaviour.)

Note that in the 'smooth' setting POSM is outperforming BSIH and, therefore, its regret is negative. Furthermore, in the considerably more difficult 'non-smooth' setting all algorithms perform badly (as expected). Nevertheless, in a slightly easier case of larger zero-zone, BSIH performs the best of the three, and POSM performance starts getting better.

While BSIH is a baseline that can not be implemented as it requires to foresee the future, POSM is a correct algorithm for dynamic difficulty adjustment. Therefore it is surprising that POSM performs almost as good as BSIH or even better.

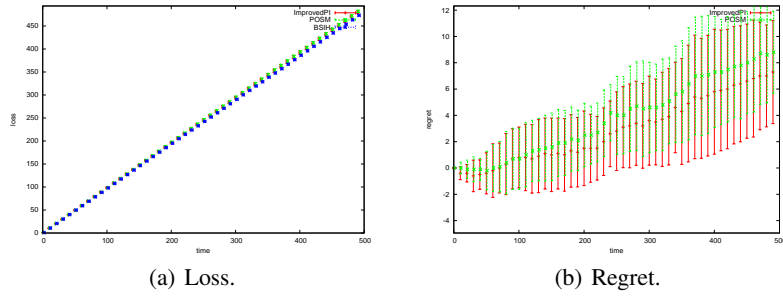

| (a) Loss. | (b) Regret. |

Figure 3: Stochastic adversary, 'non-smooth' setting, exactly one vertex in the zero-zone, on a single chain of 50 vertices.

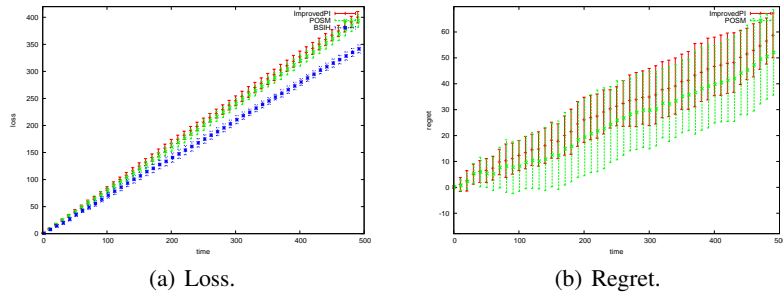

| (a) Loss. | (b) Regret. |

Figure 4: Stochastic adversary, 'non-smooth' setting, up to 20% of all vertices may be in the zero-zone, on a single chain of 50 vertices.

## 6.2 Evil Adversary

While the experiments in our stochastic environment show encouraging results, of real interest to us is the situation where the adversary is 'evil', non-stochastic, and furthermore, non-oblivious. In dynamic difficulty adjustment the algorithm will have to deal with people, who are learning and changing in hard to predict ways. We limit our experiments to a case of a linear order on difficulty settings, in other words, the chain. Even though it is a simplified scenario, this situation is rather natural for games and it demonstrates the power of our algorithm.

To simulate this situation, we've decided to use people as adversaries. Just as in dynamic difficulty adjustment players are not supposed to be aware of the mechanics, our methods and goals were not disclosed to the testing persons. Instead they were presented with a modified game of cups: On every iteration the casino is hiding a coin under one of the cups; after that the player can point at two of the cups. If the coin is under one of these two, the player wins it. Behind the scenes the cups represented the vertices on the chain and the players' choices were setting the lower and upper borders of the zero-zone. If the algorithm's prediction was wrong, one of the two cups was decided on randomly and the coin was placed under it. If the prediction was correct, no coin was awarded.

Unfortunately, using people in such experiments places severe limitations on the size of the game. In a simplified setting as this and without any extrinsic rewards they can only handle short chains and short games before getting bored. In our case we restricted the length of the chain to 8 and the length of each game to 15. It is possible to simulate a longer game by not resetting the weights of the algorithm after each game is over, but at the current stage of work it wasn't done.

Again, we created the 'smooth' and 'non-smooth' setting by placing or removing restrictions on how players were allowed to choose their cups. To each game either IMPROVEDPI or POSM was assigned. The results for the 'smooth' setting are on Figures 5(a), 5(b), and 5(c); for the 'non-smooth' on Figures 6(a), 5(b), and 6(c). Note, that due to the fact that this time different games were played by IMPROVEDPI and POSM, we have two different plots for their corresponding loss values.

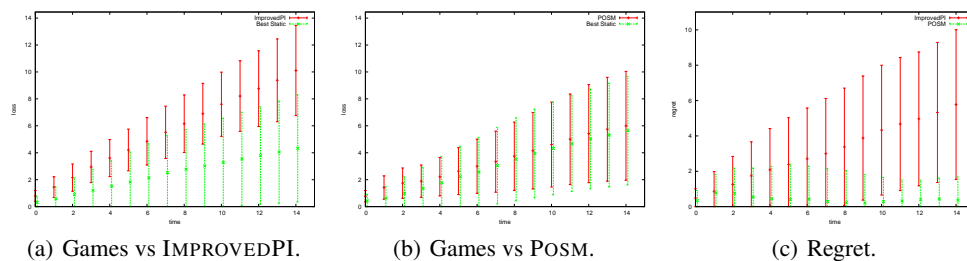

| (a) Games vs IMPROVEDPI. | (b) Games vs POSM. | (c) Regret. |

Figure 5: Evil adversary, 'smooth' setting, a single chain of 8 vertices.

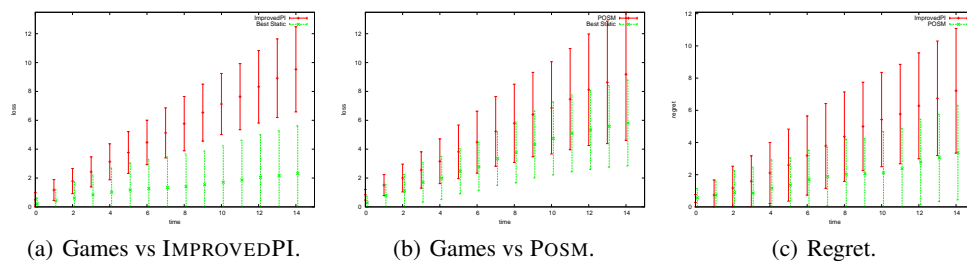

| (a) Games vs IMPROVEDPI. | (b) Games vs POSM. | (c) Regret. |

Figure 6: Evil adversary, 'non-smooth' setting, a single chain of 8 vertices.

We can see that in the 'smooth' setting again the performance of POSM is very close to that of BSIH. In the more difficult 'non-smooth' one the results are also encouraging. Note, that the loss of BSIH appears to be worse in games played by POSM. A plausible interpretation is that players had to follow more difficult (less static) strategies to fool POSM to win their coins. Nevertheless, the regret of POSM is small even in this case.

## 7  Conclusions

In this paper we formalised dynamic difficulty adjustment as a prediction problem on partially ordered sets and proposed a novel online learning algorithm, POSM, for dynamic difficulty adjustment. Using this formalisation, we were able to prove a bound on the performance of POSM relative to the best static difficulty setting chosen in hindsight, BSIH. To validate our theoretical findings empirically, we performed a set of experiments, comparing POSM and another state-of-the-art algorithm to BSIH in two settings $(a)$ simulating the player by a stochastic process and $(b)$ simulating the player by humans that are encouraged to play as adverserially as possible. These experiments showed that POSM performs very often almost as well as BSIH and, even more surprisingly, sometimes even better. As this is also even better than the behaviour suggested by our mistake bound, there seems to be a gap between the theoretical and empirical performance of our algorithm.

In future work we will on the one hand investigate this gap, aiming at providing better bounds by, perhaps, making stronger but still realistic assumptions. On the other hand, we will implement POSM in a range of computer games as well as teaching systems to observe its behaviour in real application scenarios.

## Acknowledgments

This work was supported in part by the German Science Foundation (DFG) in the Emmy Noether-program under grant 'GA 1615/1-1'. The authors thank Michael Kamp for proofreading.

# References

[1] J. Abernethy, E. Hazan, and A. Rakhlin. Competing in the dark: An efficient algorithm for bandit linear optimization. 2008.

[2] P. Auer, N. Cesa-Bianchi, Y. Freund, and R. Schapire. Gambling in a rigged casino: The adversarial multi-armed bandit problem. *Foundations of Computer Science, Annual IEEE Symposium on*, 0:322, 1995.

[3] N. Cesa-Bianchi and G. Lugosi. *Prediction, learning, and games*. Cambridge University Press, 2006.

[4] N. Cesa-Bianchi, Y. Mansour, and G. Stoltz. Improved second-order bounds for prediction with expert advice. *Machine Learning*, 66:321–352, 2007. 10.1007/s10994-006-5001-7.

[5] D. Charles and M. Black. Dynamic player modeling: A framework for player-centered digital games. In *Proc. of the International Conference on Computer Games: Artificial Intelligence, Design and Education*, pages 29–35, 2004.

[6] V. Dani and T. P. Hayes. Robbing the bandit: less regret in online geometric optimization against an adaptive adversary. In *Proceedings of the seventeenth annual ACM-SIAM symposium on Discrete algorithm*, SODA '06, pages 937–943, New York, NY, USA, 2006. ACM.

[7] G. Danzi, A. H. P. Santana, A. W. B. Furtado, A. R. Gouveia, A. Leitão, and G. L. Ramalho. Online adaptation of computer games agents: A reinforcement learning approach. *II Workshop de Jogos e Entretenimento Digital*, pages 105–112, 2003.

[8] T. Gärtner and G. C. Garriga. The cost of learning directed cuts. In *Proceedings of the 18th European Conference on Machine Learning*, 2007.

[9] R. Herbrich, T. Minka, and T. Graepel. Trueskill$^{tm}$: A bayesian skill rating system. In *NIPS*, pages 569–576, 2006.

[10] R. Hunicke and V. Chapman. AI for dynamic difficulty adjustment in games. *Proceedings of the Challenges in Game AI Workshop, Nineteenth National Conference on Artificial Intelligence*, 2004.

[11] H. McMahan and A. Blum. Online geometric optimization in the bandit setting against an adaptive adversary. In J. Shawe-Taylor and Y. Singer, editors, *Learning Theory*, volume 3120 of *Lecture Notes in Computer Science*, pages 109–123. Springer Berlin / Heidelberg, 2004.

[12] O. Missura and T. Gärtner. Player Modeling for Intelligent Difficulty Adjustment. In *Discovery Science*, pages 197–211. Springer, 2009.

[13] J. Togelius, R. Nardi, and S. Lucas. Making racing fun through player modeling and track evolution. In *SAB'06 Workshop on Adaptive Approaches for Optimizing Player Satisfaction in Computer and Physical Games*, pages 61–70, 2006.

[14] G. Yannakakis and M. Maragoudakis. Player Modeling Impact on Player's Entertainment in Computer Games. *Lecture notes in computer science*, 3538:74, 2005.

